# Non-Parametric Bayesian Dictionary Learning for Sparse Image Representations

**Mingyuan Zhou, Haojun Chen, John Paisley, Lu Ren, [1]Guillermo Sapiro and Lawrence Carin**
Department of Electrical and Computer Engineering
Duke University, Durham, NC 27708-0291, USA
[1]Department of Electrical and Computer Engineering
University of Minnesota, Minneapolis, MN 55455, USA
`{mz1,hc44,jwp4,lr,lcarin}@ee.duke.edu, {guille}@umn.edu`

## Abstract

Non-parametric Bayesian techniques are considered for learning dictionaries for sparse image representations, with applications in denoising, inpainting and compressive sensing (CS). The beta process is employed as a prior for learning the dictionary, and this non-parametric method naturally infers an appropriate dictionary size. The Dirichlet process and a probit stick-breaking process are also considered to exploit structure within an image. The proposed method can learn a sparse dictionary *in situ*; training images may be exploited if available, but they are not required. Further, the noise variance need not be known, and can be non-stationary. Another virtue of the proposed method is that sequential inference can be readily employed, thereby allowing scaling to large images. Several example results are presented, using both Gibbs and variational Bayesian inference, with comparisons to other state-of-the-art approaches.

## 1 Introduction

There has been significant recent interest in sparse signal expansions in several settings. For example, such algorithms as the support vector machine (SVM) [1], the relevance vector machine (RVM) [2], Lasso [3] and many others have been developed for sparse regression (and classification). A sparse representation has several advantages, including the fact that it encourages a simple model, and therefore over-training is often avoided. The inferred sparse coefficients also often have biological/physical meaning, of interest for model interpretation [4].

Of relevance for the current paper, there has recently been significant interest in sparse representations in the context of denoising, inpainting [5–10], compressive sensing (CS) [11, 12], and classification [13]. All of these applications exploit the fact that most images may be sparsely represented in an appropriate dictionary. Most of the CS literature assumes "off-the-shelf" wavelet and DCT bases/dictionaries [14], but recent denoising and inpainting research has demonstrated the significant advantages of learning an often over-complete dictionary matched to the signals of interest (*e.g.*, images) [5–10, 12, 15]. The purpose of this paper is to perform dictionary learning using new non-parametric Bayesian technology [16, 17], that offers several advantages not found in earlier approaches, which have generally sought point estimates.

This paper makes four main contributions:

• The dictionary is learned using a beta process construction [16, 17], and therefore the number of dictionary elements and their relative importance may be inferred non-parametrically.
• For the denoising and inpainting applications, we do not have to assume *a priori* knowledge of the noise variance (it is inferred within the inversion). The noise variance can also be non-stationary.
• The spatial inter-relationships between different components in images are exploited by use of the Dirichlet process [18] and a probit stick-breaking process [19].

• Using learned dictionaries, inferred off-line or *in situ*, the proposed approach yields CS performance that is markedly better than existing standard CS methods as applied to imagery.

## 2 Dictionary Learning with a Beta Process

In traditional *sparse coding* tasks, one considers a signal $x \in \Re^n$ and a *fixed* dictionary $\mathbf{D} = (d_1, d_2, \ldots, d_M)$ where each $d_m \in \Re^n$. We wish to impose that any $x \in \Re^n$ may be represented approximately as $\hat{x} = \mathbf{D}\alpha$, where $\alpha \in \Re^M$ is sparse, and our objective is to also minimize the $\ell_2$ error $\|\hat{x} - x\|_2$. With a proper dictionary, a sparse $\alpha$ often manifests robustness to noise (the model doesn't fit noise well), and the model also yields effective inference of $\alpha$ even when $x$ is partially or indirectly observed via a small number of measurements (of interest for inpainting, interpolation and compressive sensing [5, 7]). To the authors' knowledge, all previous work in this direction has been performed in the following manner: (*i*) if $\mathbf{D}$ is given, the sparse vector $\alpha$ is estimated via a point estimate (without a posterior distribution), typically based on orthogonal matching pursuits (OMP), basis pursuits or related methods, for which the stopping criteria is defined by assuming knowledge (or off-line estimation) of the noise variance or the sparsity level of $\alpha$; and (*ii*) when the dictionary $\mathbf{D}$ is to be learned, the dictionary size $M$ must be set *a priori*, and a point estimate is achieved for $\mathbf{D}$ (in practice one may infer $M$ via cross-validation, with this step avoided in the proposed method). In many applications one may not know the noise variance or an appropriate sparsity level of $\alpha$; further, one may be interested in the confidence of the estimate (*e.g.*, "error bars" on the estimate of $\alpha$). To address these goals, we propose development of a non-parametric Bayesian formulation to this problem, in terms of the beta process, this allowing one to infer the appropriate values of $M$ and $\|\alpha\|_0$ (sparsity level) jointly, also manifesting a full posterior density function on the learned $\mathbf{D}$ and the inferred $\alpha$ (for a particular $x$), yielding a measure of confidence in the inversion. As discussed further below, the non-parametric Bayesian formulation also allows one to relax other assumptions that have been made in the field of learning $\mathbf{D}$ and $\alpha$ for denoising, inpainting and compressive sensing. Further, the addition of other goals are readily addressed within the non-parametric Bayesian paradigm, *e.g.* designing $\mathbf{D}$ for *joint* compression *and* classification.

### 2.1 Beta process formulation

We desire the model $x = \mathbf{D}\alpha + \epsilon$, where $x \in \Re^n$ and $\mathbf{D} \in \Re^{n \times M}$, and we wish to learn $\mathbf{D}$ and in so doing infer $M$. Toward this end, we consider a dictionary $\mathbf{D} \in \Re^{n \times K}$, with $K \to \infty$; by inferring the number of columns of $\mathbf{D}$ that are required for accurate representation of $x$, the appropriate value of $M$ is implicitly inferred (work has been considered in [20, 21] for the related but distinct application of factor analysis). We wish to also impose that $\alpha \in \Re^K$ is sparse, and therefore only a small fraction of the columns of $\mathbf{D}$ are used for representation of a given $x$. Specifically, assume that we have a training set $\mathcal{D} = \{x_i, y_i\}_{i=1,N}$, where $x_i \in \Re^n$ and $y_i \in \{1, 2, \ldots, N_c\}$, where $N_c \geq 2$ represents the number of classes from which the data arise; when learning the dictionary we ignore the class labels $y_i$, and later discuss how they may be considered in the learning process.

The two-parameter beta process (BP) was developed in [17], to which the reader is referred for further details; we here only provide those details of relevance for the current application. The BP with parameters $a > 0$ and $b > 0$, and base measure $H_0$, is represented as $\mathrm{BP}(a, b, H_0)$, and a draw $H \sim \mathrm{BP}(a, b, H_0)$ may be represented as

$$H(\psi) = \sum_{k=1}^{K} \pi_k \delta_{\psi_k}(\psi) \qquad \pi_k \sim \mathrm{Beta}(a/K, b(K-1)/K) \qquad \psi_k \sim H_0 \qquad (1)$$

with this a valid measure as $K \to \infty$. The expression $\delta_{\psi_k}(\psi)$ equals one if $\psi = \psi_k$ and is zero otherwise. Therefore, $H(\psi)$ represents a vector of $K$ probabilities, with each associated with a respective atom $\psi_k$. In the limit $K \to \infty$, $H(\psi)$ corresponds to an infinite-dimensional vector of probabilities, and each probability has an associated atom $\psi_k$ drawn i.i.d. from $H_0$.

Using $H(\psi)$, we may now draw $N$ *binary* vectors, the $i$th of which is denoted $z_i \in \{0, 1\}^K$, and the $k$th component of $z_i$ is drawn $z_{ik} \sim \mathrm{Bernoulli}(\pi_k)$. These $N$ binary column vectors are used to constitute a matrix $\mathbf{Z} \in \{0, 1\}^{K \times N}$, with $i$th column corresponding to $z_i$; the $k$th row of $\mathbf{Z}$ is associated with atom $\psi_k$, drawn as discussed above. For our problem the atoms $\psi_k \in \Re^n$ will correspond to candidate members of our dictionary $\mathbf{D}$, and the binary vector $z_i$ defines which members of the dictionary are used to represent sample $x_i \in \mathcal{D}$.

Let $\boldsymbol{\Psi} = (\boldsymbol{\psi}_1, \boldsymbol{\psi}_2, \ldots, \boldsymbol{\psi}_K)$, and we may consider the limit $K \to \infty$. A naive form of our model, for representation of sample $\boldsymbol{x}_i \in \mathcal{D}$, is $\boldsymbol{x}_i = \boldsymbol{\Psi} \boldsymbol{z}_i + \boldsymbol{\epsilon}_i$. However, this is highly restrictive, as it imposes that the coefficients of the dictionary expansion must be binary. To address this, we draw weights $\boldsymbol{w}_i \sim \mathcal{N}(0, \gamma_w^{-1} \mathbf{I}_K)$, where $\gamma_w$ is the precision or inverse variance; the dictionary weights are now $\boldsymbol{\alpha}_i = \boldsymbol{z}_i \circ \boldsymbol{w}_i$, and $\boldsymbol{x}_i = \boldsymbol{\Psi} \boldsymbol{\alpha}_i + \boldsymbol{\epsilon}_i$, where $\circ$ represents the Hadamard (element-wise) multiplication of two vectors. Note that, by construction, $\boldsymbol{\alpha}$ is sparse; this imposition of sparseness is distinct from the widely used Laplace shrinkage prior [3], which imposes that many coefficients are small but not necessarily exactly zero.

For simplicity we assume that the dictionary elements, defined by the atoms $\boldsymbol{\psi}_k$, are drawn from a multivariate Gaussian base $H_0$, and the components of the error vectors $\boldsymbol{\epsilon}_i$ are drawn i.i.d. from a zero-mean Gaussian. The hierarchical form of the model may now be expressed as

$$
\begin{aligned}
\boldsymbol{x}_i &= \boldsymbol{\Psi} \boldsymbol{\alpha}_i + \boldsymbol{\epsilon}_i \,, & \boldsymbol{\alpha}_i &= \boldsymbol{z}_i \circ \boldsymbol{w}_i \\
\boldsymbol{\Psi} &= (\boldsymbol{\psi}_1, \boldsymbol{\psi}_2, \ldots, \boldsymbol{\psi}_K) \,, & \boldsymbol{\psi}_k &\sim \mathcal{N}(0, n^{-1} \mathbf{I}_n) \\
\boldsymbol{w}_i &\sim \mathcal{N}(0, \gamma_w^{-1} \mathbf{I}_K) \,, & \boldsymbol{\epsilon}_i &\sim \mathcal{N}(0, \gamma_\epsilon^{-1} \mathbf{I}_n) \\
\boldsymbol{z}_i &\sim \prod_{k=1}^{K} \mathrm{Bernoulli}(\pi_k) \,, & \pi_k &\sim \mathrm{Beta}(a/K, b(K-1)/K)
\end{aligned}
\tag{2}
$$

Non-informative gamma hyper-priors are typically placed on $\gamma_w$ and $\gamma_\epsilon$. Consecutive elements in the above hierarchical model are in the conjugate exponential family, and therefore inference may be implemented via a variational Bayesian [22] or Gibbs-sampling analysis, with analytic update equations (all inference update equations, and the software, can be found at *http://people.ee.duke.edu/~lihan/cs/*). After performing such inference, we retain those columns of $\boldsymbol{\Psi}$ that are used in the representation of the data in $\mathcal{D}$, thereby inferring $\mathbf{D}$ and hence $M$.

To impose our desire that the vector of dictionary weights $\boldsymbol{\alpha}$ is sparse, one may adjust the parameters $a$ and $b$. Particularly, as discussed in [17], in the limit $K \to \infty$, the number of elements of $\boldsymbol{z}_i$ that are non-zero is a random variable drawn from $\mathrm{Poisson}(a/b)$. In Section 3.1 we discuss the fact that these parameters are in general non-informative and the sparsity is intrinsic to the data.

## 2.2 Accounting for a classification task

There are problems for which it is desired that $\boldsymbol{x}$ is sparsely rendered in $\mathbf{D}$, and the associated weight vector $\boldsymbol{\alpha}$ may be employed for other purposes beyond representation. For example, one may perform a classification task based on $\boldsymbol{\alpha}$. If one is interested in *joint* compression and classification, both goals should be accounted for when designing $\mathbf{D}$. For simplicity, we assume that the number of classes is $N_C = 2$ (binary classification), with this readily extended [23] to $N_C > 2$.

Following [9], we may define a linear or bilinear classifier based on the sparse weights $\boldsymbol{\alpha}$ and the associated data $\boldsymbol{x}$ (in the bilinear case), with this here implemented in the form of a probit classifier. We focus on the linear model, as it is simpler (has fewer parameters), and the results in [9] demonstrated that it was often as good or better than the bilinear classifier. To account for classification, the model in (2) remains unchanged, and the following may be added to the top of the hierarchy: $y_i = 1$ if $\boldsymbol{\theta}^T \hat{\boldsymbol{\alpha}} + \nu > 0$, $y_i = 2$ if $\boldsymbol{\theta}^T \hat{\boldsymbol{\alpha}} + \nu < 0$, $\boldsymbol{\theta} \sim \mathcal{N}(0, \gamma_\theta^{-1} \mathbf{I}_{K+1})$, and $\nu \sim \mathcal{N}(0, \gamma_0^{-1})$, where $\hat{\boldsymbol{\alpha}} \in \Re^{K+1}$ is the same as $\boldsymbol{\alpha} \in \Re^K$ with an appended one, to account for the classifier bias. Again, one typically places (non-informative) gamma hyper-priors on $\gamma_\theta$ and $\gamma_0$. With the added layers for the classifier, the conjugate-exponential character of the model is retained, sustaining the ability to perform VB or MCMC inference with analytic update equations. Note that the model in (2) may be employed for unlabeled data, and the extension above may be employed for the available labeled data; consequently, all data (labeled and unlabeled) may be processed jointly to infer $\mathbf{D}$.

## 2.3 Sequential dictionary learning for large training sets

In the above discussion, we implicitly assumed all data $\mathcal{D} = \{\boldsymbol{x}_i, y_i\}_{i=1,N}$ are used together to infer the dictionary $\mathbf{D}$. However, in some applications $N$ may be large, and therefore such a "batch" approach is undesirable. To address this issue one may partition the data as $\mathcal{D} = \mathcal{D}_1 \cup \mathcal{D}_2 \cup \ldots \mathcal{D}_{J-1} \cup \mathcal{D}_J$, with the data processed sequentially. This issue has been considered for point estimates of $\mathbf{D}$ [8], in which considerations are required to assure algorithm convergence. It is of interest to briefly note that sequential inference is handled naturally via the proposed Bayesian analysis.

Specifically, let $p(\mathbf{D}|\mathcal{D}, \boldsymbol{\Theta})$ represent the posterior on the desired dictionary, with all other model parameters marginalized out (*e.g.*, the sample-dependent coefficients $\boldsymbol{\alpha}$); the vector $\boldsymbol{\Theta}$ represents the model hyper-parameters. In a Bayesian analysis, rather than evaluating $p(\mathbf{D}|\mathcal{D}, \boldsymbol{\Theta})$ directly, one may employ the same model (prior) to infer $p(\mathbf{D}|\mathcal{D}_1, \boldsymbol{\Theta})$. This posterior may then serve as a prior for $\mathbf{D}$ when considering next $\mathcal{D}_2$, inferring $p(\mathbf{D}|\mathcal{D}_1 \cup \mathcal{D}_2, \boldsymbol{\Theta})$. When doing variational Bayesian (VB) inference we have an analytic approximate representation for posteriors such as $p(\mathbf{D}|\mathcal{D}_1, \boldsymbol{\Theta})$, while for Gibbs sampling we may use the inferred samples. When presenting results in Section 5, we discuss additional means of sequentially accelerating a Gibbs sampler.

## 3 Denoising, Inpainting and Compressive Sensing

### 3.1 Image Denoising and Inpainting

Assume we are given an image $\mathbf{I} \in \Re^{N_y \times N_x}$ with additive noise and missing pixels; we here assume a monochrome image for simplicity, but color images are also readily handled, as demonstrated when presenting results. As is done typically [6, 7], we partition the image into $N_B = (N_y - B + 1) \times (N_x - B + 1)$ overlapping blocks $\{\boldsymbol{x}_i\}_{i=1,N_B}$, for each of which $\boldsymbol{x}_i \in \Re^{B^2}$ ($B = 8$ is typically used). If there is only additive noise but no missing pixels, then the model in (2) can be readily applied for simultaneous dictionary learning and image denoising. If there are both noise and missing pixels, instead of directly observing $\boldsymbol{x}_i$, we observe a subset of the pixels in each $\boldsymbol{x}_i$. Note that here $\boldsymbol{\Psi}$ and $\{\boldsymbol{\alpha}_i\}_{i=1,N_B}$, which are used to recover the original noise-free and complete image, are directly inferred from the data under test; one may also employ an appropriate training set $\mathcal{D}$ with which to learn a dictionary $\mathbf{D}$ offline, or for initialization of *in situ* learning.

In denoising and inpainting studies of this type (see for example [6, 7] and references therein), it is often assumed that either the variance is known and used as a "stopping" criteria, or that the sparsity level is pre-determined and fixed for all $i \in \{1, N_B\}$. While these may be practical in some applications, we feel it is more desirable to not make these assumptions. In (2) the noise precision (inverse variance), $\gamma_\epsilon$, is assumed drawn from a non-informative gamma distribution, and a full posterior density function is inferred for $\gamma_\epsilon$ (and all other model parameters). In addition, the problems of addressing spatially nonuniform noise as well as nonuniform noise across color channels are of interest [7]; they are readily handled in the proposed model by drawing a separate precision $\gamma_\epsilon$ for each color channel in each $B \times B$ block, each of which is drawn from a shared gamma prior.

The sparsity level of the representation in our model, *i.e.*, $\{\|\boldsymbol{\alpha}_i\|_0\}_{i=1,N}$, is influenced by the parameters $a$ and $b$ in the beta prior in (2). Examining the posterior $p(\pi_k|-) \sim Beta(a/K + \sum_{i=1}^N z_{ik}, b(K-1)/K + N - \sum_{i=1}^N z_{ik})$, conditioned on all other parameters, we find that most settings of $a$ and $b$ tend to be non-informative, especially in the case of sequential learning (discussed further in Section 5). Therefore, the average sparsity level of the representation is inferred by the data itself and each sample $\boldsymbol{x}_i$ has its own unique sparse representation based on the posterior, which renders much more flexibility than enforcing the same sparsity level for each sample.

### 3.2 Compressive sensing

We consider CS in the manner employed in [12]. Assume our objective is to measure an image $\mathbf{I} \in \Re^{N_y \times N_x}$, with this image constituting the $8 \times 8$ blocks $\{\boldsymbol{x}_i\}_{i=1,N_B}$. Rather than measuring the $\boldsymbol{x}_i$ directly, pixel-by-pixel, in CS we perform the projection measurement $\boldsymbol{v}_i = \boldsymbol{\Phi}\boldsymbol{x}_i$, where $\boldsymbol{v}_i \in \Re^{N_p}$, with $N_p$ representing the number of projections, and $\boldsymbol{\Phi} \in \Re^{N_p \times 64}$ (assuming that $\boldsymbol{x}_i$ is represented by a 64-dimensional vector). There are many (typically random) ways in which $\boldsymbol{\Phi}$ may be constructed, with the reader referred to [24]. Our goal is to have $N_p \ll 64$, thereby yielding compressive measurements. Based on the CS measurements $\{\boldsymbol{v}_i\}_{i=1,N_B}$, our objective is to recover $\{\boldsymbol{x}_i\}_{i=1,N_B}$.

Consider a potential dictionary $\boldsymbol{\Psi}$, as discussed in Section 2. It is assumed that for each of the $\{\boldsymbol{x}_i\}_{i=1,N_B}$ from the image under test $\boldsymbol{x}_i = \boldsymbol{\Psi}\boldsymbol{\alpha}_i + \epsilon_i$, for sparse $\boldsymbol{\alpha}_i$ and relatively small error $\|\epsilon_i\|_2$. The number of required projections $N_p$ needed for accurate estimation of $\boldsymbol{\alpha}_i$ is proportional to $\|\boldsymbol{\alpha}_i\|_0$ [11], with this underscoring the desirability of learning a dictionary in which very sparse representations are manifested (as compared to using an "off-the-shelf" wavelets or DCT basis).

For CS inversion, the model in (2) is employed, and therefore the appropriate dictionary $\mathbf{D}$ is learned *jointly* while performing CS inversion, *in situ* on the image under test. When performing CS analy-

sis, in (2), rather than observing $\boldsymbol{x}_i$, we observe $\boldsymbol{v}_i = \boldsymbol{\Phi}\mathbf{D}\boldsymbol{\alpha}_i + \boldsymbol{\epsilon}_i$, for $i = 1, \ldots, N_B$ (the likelihood function is therefore modified slightly).

As discussed when presenting results, one may also learn the CS dictionary in advance, off-line, with appropriate training images (using the model in (2)). However, the unique opportunity for *joint* CS inversion and learning of an appropriate parsimonious dictionary is deemed to be a significant advantage, as it does not presuppose that one would know an appropriate training set in advance.

The inpainting problem may be viewed as a special case of CS, in which each row of $\boldsymbol{\Phi}$ corresponds to a delta function, locating a unique pixel on the image at which useful (unobscured) data are observed. Those pixels that are unobserved, or that are contaminated (*e.g.*, by superposed text [7]) are not considered when inferring the $\boldsymbol{\alpha}_i$ and $\mathbf{D}$. A CS camera designed around an inpainting construction has several advantages, from the standpoint of simplicity. As observed from the results in Section 5, an inpainting-based CS camera would simply observe a subset of the usual pixels, selected at random.

## 4   Exploiting Spatial Structure

For the applications discussed above, the $\{\boldsymbol{x}_i\}_{i=1,N_B}$ come from the single image under test, and consequently there is underlying (spatial) structure that should ideally be exploited. Rather than re-writing the entire model in (2), we focus on the following equations in the hierarchy: $\boldsymbol{z}_i \sim \prod_{k=1}^{K} \text{Bernoulli}(\pi_k)$, and $\boldsymbol{\pi} \sim \prod_{k=1}^{K} \text{Beta}(a/K, b(K-1)/K)$. Instead of having a single vector $\boldsymbol{\pi} = \{\pi_1, \ldots, \pi_K\}$ that is shared for all $\{\boldsymbol{x}_i\}_{i=1,N_B}$, it is expected that there may be a mixture of $\boldsymbol{\pi}$ vectors, corresponding to different segments in the image. Since the number of mixture components is not known *a priori*, this mixture model is modeled via a Dirichlet process [18]. We may therefore employ, for $i = 1, \ldots, N_B$,

$$\boldsymbol{z}_i \sim \prod_{k=1}^{K} \text{Bernoulli}(\pi_{ik}) \qquad \boldsymbol{\pi}_i \sim G \qquad G \sim \text{DP}(\beta, \prod_{k=1}^{K} \text{Beta}(a/K, b(K-1)/K)) \tag{3}$$

Alternatively, we may cluster the $\boldsymbol{z}_i$ directly, yielding $\boldsymbol{z}_i \sim G$, $G \sim \text{DP}(\beta, \prod_{k=1}^{K} \text{Bernoulli}(\pi_k))$, $\boldsymbol{\pi} \sim \prod_{k=1}^{K} \text{Beta}(a/K, b(K-1)/K)$, where the $\boldsymbol{z}_i$ are drawn i.i.d. from $G$. In practice we implement such DP constructions via a truncated stick-breaking representation [25], again retaining the conjugate-exponential structure of interest for analytic VB or Gibbs inference. In such an analysis we place a non-informative gamma prior on the precision $\beta$.

The construction in (3) clusters the blocks, and therefore it imposes structure not constituted in the simpler model in (2). However, the DP still assumes that the members of $\{\boldsymbol{x}_i\}_{i=1,N_B}$ are exchangeable. Space limitations preclude discussing this matter in detail here, but we have also considered replacement of the DP framework above with a probit stick-breaking process (PSBP) [19], which explicitly imposes that it is more likely for proximate blocks to be in the same cluster, relative to distant blocks. When presenting results, we show examples in which PSBP has been used, with its relative effectiveness compared to the simpler DP construction. The PSBP again retains full conjugate-exponential character within the hierarchy, of interest for efficient inference, as discussed above.

## 5   Example Results

For the denoising and inpainting results, we observed that the Gibbs sampler provided better performance than associated variational Bayesian inference. For denoising and inpainting we may exploit shifted versions of the data, which accelerates convergence substantially (discussed in detail below). Therefore, all denoising and inpainting results are based on efficient Gibbs sampling. For CS we cannot exploit shifted images, and therefore to achieve fast inversion variational Bayesian (VB) inference [22] is employed; for this application VB has proven to be quite effective, as discussed below. The same set of model hyper-parameters are used across all our denoising, inpainting and CS examples (no tuning was performed): all gamma priors are set as $\text{Gamma}(10^{-6}, 10^{-6})$, along the lines suggested in [2], and the beta distribution parameters are set with $a = K$ and $b = N/8$ (many other settings of $a$ and $b$ yield similar results).

## 5.1 Denoising

We consider denoising a $256 \times 256$ image, with comparison of the proposed approach to K-SVD [6] (for which the noise variance is assumed known and fixed); the *true* noise standard deviation is set at 15, 25 and 50 in the examples below. We show results for three algorithms: ($i$) mismatched K-SVD (with noise standard deviation of 30), ($ii$) K-SVD when the standard deviation is properly matched, and ($iii$) the proposed BP approach. For ($iii$) a non-informative prior is placed on the noise precision, and the same BP model is run for all three noise levels (with the underlying noise levels inferred). The BP and K-SVD employed no *a priori* training data. In Figure 1 are shown the noisy images at the three different noise levels, as well as the reconstructions via BP and K-SVD. A preset large dictionary size $K = 256$ is used for both algorithms, and for the BP results we inferred that approximately M = 196, 128, and 34 dictionary elements were important for noise standard deviations 15, 25, and 50, respectively; the remaining elements of the dictionary were used less than 0.1% of the time. As seen within the bottom portion of the right part of Figure 1, the unused dictionary elements appear as random draws from the prior, since they are not used and hence influenced by the data.

Note that K-SVD works well when the set noise variance is at or near truth, but the method is undermined by mismatch. The proposed BP approach is robust to changing noise levels. Quantitative performance is summarized in Table 1. The BP denoiser estimates a full posterior density function on the noise standard deviation; for the examples considered here, the modes of the inferred standard-deviation posteriors were 15.57, 25.35, and 48.12, for true standard deviations 15, 25, and 50, respectively.

To achieve these BP results, we employ a sequential implementation of the Gibbs sampler (a batch implementation converges to the same results but with higher computational cost); this is discussed in further detail below, when presenting inpainting results.

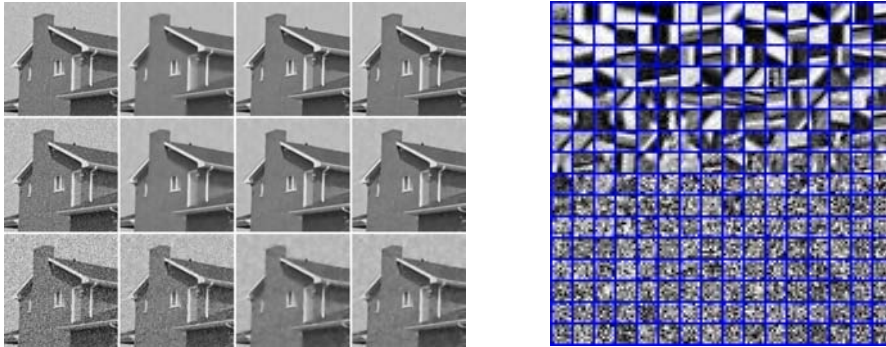

Figure 1: Left: Representative denoising results, with the top through bottom rows corresponding to noise standard deviations of 15, 25 and 50, respectively. The second and third columns represent K-SVD [6] results with assumed standard deviation equal to 30 and the ground truth, respectively. The fourth column represents the proposed BP reconstructions. The noisy images are in the first column. Right: Inferred BP dictionary elements for noise standard deviation 25, in order of importance (probability to be used) from the top-left.

Table 1: Peak signal-to-reconstructed image measure (PSNR) for the data in Figure 1, for K-SVD [6] and the proposed BP method. The true standard deviation was 15, 25 and 50, respectively, from the top to the bottom row. For the mismatched K-SVD results, the noise stand deviation was fixed at 30.

| Original Noisy Image (dB) | K-SVD Denoising mismatched variance (dB) | K-SVD Denoising matched variance (dB) | Beta Process Denoising (dB) |
|---|---|---|---|
| 24.58 | 30.67 | 34.32 | 34.44 |
| 20.19 | 31.52 | 32.15 | 32.17 |
| 14.56 | 19.60 | 27.95 | 28.08 |

## 5.2 Inpainting

Our inpainting and denoising results were achieved by using the following sequential procedure. Consider any pixel $[p, j]$, where $p, j \in [1, B]$, and let this pixel constitute the left-bottom pixel in a new $B \times B$ block. Further, consider all $B \times B$ blocks with left-bottom pixels at $\{p + \ell B, j +$

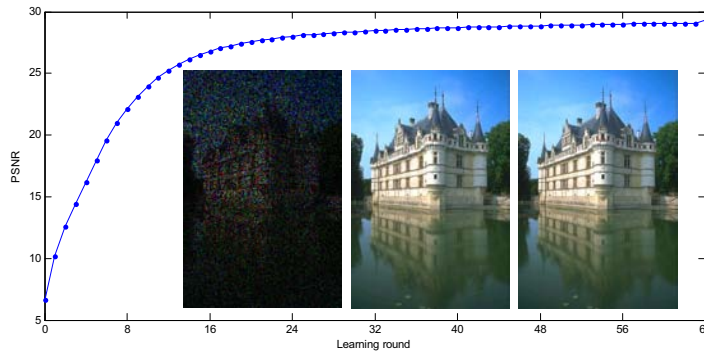

Figure 2: Inpainting results. The curve shows the PSNR as a function of the $B^2 = 64$ Gibbs learning rounds. The left figure is the test image, with $80\%$ of the RGB pixels missing, the middle figure is the result after 64 after Gibbs rounds (final result), and the right figure is the original uncontaminated image.

$mB\} \cup \delta(p-1)\{N_y - B + 1, j + mB\} \cup \delta(j-1)\{p + \ell B, N_x - B + 1\}$ for $\ell$ and $m$ that satisfy $p + \ell B \leq N_y - B + 1$ and $j + mB \leq N_x - B + 1$. This set of blocks is denoted data set $\mathcal{D}_{pj}$, and considering $1 \leq p \leq B$ and $1 \leq j \leq B$, there are a total of $B^2$ such shifted data sets. In the first iteration of learning $\boldsymbol{\Psi}$, we employ the blocks in $\mathcal{D}_{11}$, and for this first round we initialize $\boldsymbol{\Psi}$ and $\boldsymbol{\alpha}_i$ based on a singular value decomposition (SVD) of the blocks in $\mathcal{D}_{11}$ (we achieved similar results when $\boldsymbol{\Psi}$ was initialized randomly). We do several Gibbs iterations with $\mathcal{D}_{11}$ and then stop the Gibbs algorithm, retaining the last sample of $\boldsymbol{\Psi}$ and $\boldsymbol{\alpha}_i$ from the previous step. These $\boldsymbol{\Psi}$ and $\boldsymbol{\alpha}_i$ are then used to initialize the Gibbs sampler in the second round, now applied to the $B \times B$ blocks in $\mathcal{D}_{11} \cup \mathcal{D}_{21}$ (for $\mathcal{D}_{21}$ the neighboring $\boldsymbol{\alpha}_i$ is used for initialization). The Gibbs sampler is now run on this expanded data for several iterations, the last sample is retained, and the data set is augmented again. This is done $B^2 = 64$ times until at the end all shifted blocks are processed simultaneously. This sequential process may be viewed as a sequential Gibbs burn in, after which all of the shifted blocks are processed.

Theoretically, one would expect to need thousands of Gibbs iterations to achieve convergence. However, our experience is that even a *single* iteration in each of the above $B^2$ rounds yields good results. In Figure 2 we show the PSNR as a function of each of the $B^2 = 64$ rounds discussed above. For Gibbs rounds 16, 32 and 64 the corresponding PSNR values were 26.78 dB, 28.46 dB and 29.31 dB. For this example we used $K = 256$. This example was considered in [7] (we obtained similar results for the "New Orleans" image, also considered in [7]); the best results reported there were a PSNR of 29.65 dB. However, to achieve those results a training data set was employed for initialization [7]; the BP results are achieved with no *a priori* training data. Concerning computational costs, the inpainting and denoising algorithms scale linearly as a function of the block size, the dictionary size, the sparsity level, and the number of training samples; all results reported here were run efficiently in Matlab on PCs, with comparable costs as K-SVD.

## 5.3 Compressive sensing

We consider a CS example, in which the image is divided into $8 \times 8$ patches, with these constituting the underlying data $\{\boldsymbol{x}_i\}_{i=1,N_B}$ to be inferred. For each of the $N_B$ blocks, a vector of CS measurements $\boldsymbol{v}_i = \boldsymbol{\Phi}\boldsymbol{x}_i$ is measured, where the number of projections per patch is $N_p$, and the total number of CS projections is $N_p N_B$. In this example the elements of $\boldsymbol{\Phi}$ were constructed randomly as draws from $\mathcal{N}(0, 1)$, but many other projection classes may be considered [11, 24]. Each $\boldsymbol{x}_i$ is assumed represented in terms of a dictionary $\boldsymbol{x}_i = \mathbf{D}\boldsymbol{\alpha}_i + \boldsymbol{\epsilon}_i$, and three constructions for $\mathbf{D}$ were considered: (*i*) a DCT expansion; (*ii*) learning of $\mathbf{D}$ using the beta process construction, using training images; (*iii*) using the beta process to perform joint CS inversion and learning of $\mathbf{D}$. For (*ii*), the training data consisted of 4000 $8 \times 8$ patches chosen at random from 100 images selected from the Microsoft database (*http://research.microsoft.com/en-us/projects/objectclassrecognition*). The dictionary was set to $K = 256$, and the offline beta process inferred a dictionary of size $M = 237$.

Representative CS reconstruction results are shown in Figure 3, for a gray-scale version of the "castle" image. The inversion results at left are based on a learned dictionary; except for the "online BP" results, all of these results employ the same dictionary $\mathbf{D}$ learned off-line as above, and the algorithms are distinguished by different ways of estimating $\{\boldsymbol{\alpha}_i\}_{i=1,N_B}$. A range of CS-inversion

algorithms are considered from the literature, and several BP-based constructions are considered as well for CS inversion. The online BP results are quite competitive with those inferred off-line.

One also notes that the results based on a learned dictionary (left in Figure 3) are markedly better than those based on the DCT (right in Figure 3); similar results were achieved when the DCT was replaced by a wavelet representation. For the DCT-based results, note that the DP- and PSBP-based BP CS inversion results are significantly better than those of all other CS inversion algorithms. The results reported here are consistent with tests we performed using over 100 images from the aforementioned Microsoft database, not reported here in detail for brevity.

Note that CS inversion using the DP-based BP algorithm (as discussed in Section 4) yield the best results, significantly better than BP results not based on the DP, and better than all competing CS inversion algorithms (for both learned dictionaries and the DCT). The DP-based results are very similar to those generated by the probit stick-breaking process (PSBP) [19], which enforces spatial information more explicitly; this suggests that the simpler DP-based results are adequate, at least for the wide class of examples considered. Note that we also considered the DP and PSBP for the denoising and inpaiting examples above (those results were omitted, for brevity). The DP and PSBP denoising and inpainting results were similar to BP results without DP/PSBP (those presented above); this is attributed to the fact that when performing denoising/inpainting we may consider many shifted versions of the same image (as discussed when presenting the inpainting results).

Concerning computational costs, all CS inversions were run efficiently on PCs, with the specifics computational times dictated by the detailed Matlab implementation and the machine run on. A rough ranking of the computational speeds, from fastest to slowest, is as follows: StOMP-CFAR, Fast BCS, OMP, BP, LARS/Lasso, Online BP, DP BP, PSBP BP, VB BCS, Basis Pursuit; in this list, algorithms BP through Basis Pursuits have approximately the same computational costs. The DP-based BP CS inversion algorithm scales as $O(N_B \cdot N_p \cdot B^2)$.

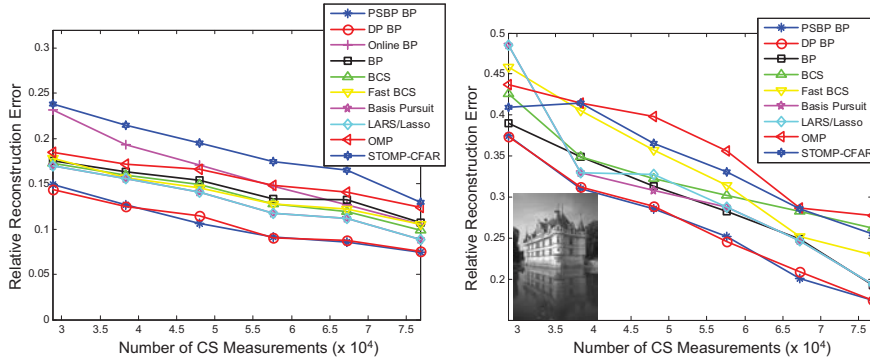

Figure 3: CS performance (fraction of $\ell_2$ error) based on learned dictionaries (left) and based on the DCT (right). For the left results, the "Online BP" results simultaneously learned the dictionary and did CS inversion; the remainder of the left results are based on a dictionary learned offline on a training set. A DCT dictionary is used for the results on the right. The underlying image under test is shown at right. Matlab code for Basis Pursuit, LARS/Lasso, OMP, STOMP are available at *http://sparselab.stanford.edu/*, and code for BCS and Fast BCS are available at *http://people.ee.duke.edu/~lihan/cs/*. The horizontal axis represents the *total* number of CS projections, $N_p N_B$. The total number of pixels in the image is $480 \times 320 = 153,600$. 99.9% of the signal energy is contained in $33,500$ DCT coefficients.

## 6   Conclusions

The non-parametric beta process has been presented for dictionary learning with the goal of image denoising, inpainting and compressive sensing, with very encouraging results relative to the state of the art. The framework may also be applied to joint compression-classification tasks. In the context of noisy underlying data, the noise variance need not be known in advance, and it need not be spatially uniform. The proposed formulation also allows unique opportunities to leverage known structure in the data, such as relative spatial locations within an image; this framework was used to achieve marked improvements in CS-inversion quality.

### Acknowledgement

The research reported here was supported in part by ARO, AFOSR, DOE, NGA and ONR.

# References

[1] N. Cristianini and J. Shawe-Taylor. *An Introduction to Support Vector Machines*. Cambridge University Press, 2000.

[2] M. Tipping. Sparse Bayesian learning and the relevance vector machine. *Journal of Machine Learning Research*, 1, 2001.

[3] R. Tibshirani. Regression shrinkage and selection via the lasso. *Journal of the Royal Statistical Society, Series B*, 58, 1994.

[4] B.A. Olshausen and D. J. Field. Sparse coding with an overcomplete basis set: A strategy employed by V1? *Vision Research*, 37, 1998.

[5] M. Aharon, M. Elad, and A. M. Bruckstein. K-SVD: An algorithm for designing overcomplete dictionaries for sparse representation. *IEEE Trans. Signal Processing*, 54, 2006.

[6] M. Elad and M. Aharon. Image denoising via sparse and redundant representations over learned dictionaries. *IEEE Trans. Image Processing*, 15, 2006.

[7] J. Mairal, M. Elad, and G. Sapiro. Sparse representation for color image restoration. *IEEE Trans. Image Processing*, 17, 2008.

[8] J. Mairal, F. Bach, J. Ponce, and G. Sapiro. Online dictionary learning for sparse coding. In *Proc. International Conference on Machine Learning*, 2009.

[9] J. Mairal, F. Bach, J. Ponce, G. Sapiro, and A. Zisserman. Supervised dictionary learning. In *Proc. Neural Information Processing Systems*, 2008.

[10] M. Ranzato, C. Poultney, S. Chopra, and Y. Lecun. Efficient learning of sparse representations with an energy-based model. In *Proc. Neural Information Processing Systems*, 2006.

[11] E. Candès and T. Tao. Near-optimal signal recovery from random projections: universal encoding strategies? *IEEE Trans. Information Theory*, 52, 2006.

[12] J.M. Duarte-Carvajalino and G. Sapiro. Learning to sense sparse signals: Simultaneous sensing matrix and sparsifying dictionary optimization. *IMA Preprint Series 2211*, 2008.

[13] J. Wright, A.Y. Yang, A. Ganesh, S.S. Sastry, and Y. Ma. Robust face recognition via sparse representation. *IEEE Trans. Pattern Analysis Machine Intelligence*, 31, 2009.

[14] S. Ji, Y. Xue, and L. Carin. Bayesian compressive sensing. *IEEE Trans. Signal Processing*, 56, 2008.

[15] R. Raina, A. Battle, H. Lee, B. Packer, and A.Y. Ng. Self-taught learning: transfer learning from unlabeled data. In *Proc. International Conference on Machine Learning*, 2007.

[16] R. Thibaux and M.I. Jordan. Hierarchical beta processes and the indian buffet process. In *Proc. International Conference on Artificial Intelligence and Statistics*, 2007.

[17] J. Paisley and L. Carin. Nonparametric factor analysis with beta process priors. In *Proc. International Conference on Machine Learning*, 2009.

[18] T. Ferguson. A Bayesian analysis of some nonparametric problems. *Annals of Statistics*, 1, 1973.

[19] A. Rodriguez and D.B. Dunson. Nonparametric bayesian models through probit stickbreaking processes. *Univ. California Santa Cruz Technical Report*, 2009.

[20] D. Knowles and Z. Ghahramani. Infinite sparse factor analysis and infinite independent components analysis. In *Proc. International Conference on Independent Component Analysis and Signal Separation*, 2007.

[21] P. Rai and H. Daumé III. The infinite hierarchical factor regression model. In *Proc. Neural Information Processing Systems*, 2008.

[22] M.J. Beal. *Variational Algorithms for Approximate Bayesian Inference*. PhD thesis, Gatsby Computational Neuroscience Unit, University College London, 2003.

[23] M. Girolami and S. Rogers. Variational Bayesian multinomial probit regression with Gaussian process priors. *Neural Computation*, 18, 2006.

[24] R.G. Baraniuk. Compressive sensing. *IEEE Signal Processing Magazine*, 24, 2007.

[25] J. Sethuraman. A constructive definition of Dirichlet priors. *Statistica Sinica*, 4, 1994.

